# Multi-armed bandits on implicit metric spaces

**Aleksandrs Slivkins**
Microsoft Research Silicon Valley
Mountain View, CA 94043
slivkins at microsoft.com

## Abstract

The multi-armed bandit (MAB) setting is a useful abstraction of many online learning tasks which focuses on the trade-off between exploration and exploitation. In this setting, an online algorithm has a fixed set of alternatives ("arms"), and in each round it selects one arm and then observes the corresponding reward. While the case of small number of arms is by now well-understood, a lot of recent work has focused on multi-armed bandits with (infinitely) many arms, where one needs to assume extra structure in order to make the problem tractable. In particular, in the Lipschitz MAB problem there is an underlying similarity metric space, known to the algorithm, such that any two arms that are close in this metric space have similar payoffs. In this paper we consider the more realistic scenario in which the metric space is *implicit* – it is defined by the available structure but not revealed to the algorithm directly. Specifically, we assume that an algorithm is given a tree-based classification of arms. For any given problem instance such a classification implicitly defines a similarity metric space, but the numerical similarity information is not available to the algorithm. We provide an algorithm for this setting, whose performance guarantees (almost) match the best known guarantees for the corresponding instance of the Lipschitz MAB problem.

## 1 Introduction

In a multi-armed bandit (MAB) problem, a *player* is presented with a sequence of trials. In each round, the player chooses one alternative from a set of alternatives ("*arms*") based on the past history, and receives the payoff associated with this alternative. The goal is to maximize the total payoff of the chosen arms. The multi-armed bandit setting was introduced in 1950s and has since been studied intensively since then in Operations Research, Economics and Computer Science, e.g. see [8] for background. This setting is often used to model the tradeoffs between exploration and exploitation, which is the principal issue in sequential decision-making under uncertainty.

One standard way to evaluate the performance of a multi-armed bandit algorithm is *regret*, defined as the difference between the expected payoff of an optimal arm and that of the algorithm. By now the multi-armed bandit problem with a small finite number of arms is quite well understood (e.g. see [22, 3, 2]). However, if the set of arms is exponentially or infinitely large, the problem becomes intractable, unless we make further assumptions about the problem instance. Essentially, an MAB algorithm needs to find a needle in a haystack; for each algorithm there are inputs on which it performs as badly as random guessing.

The bandit problems with large sets of arms have received a considerable attention, e.g. [1, 5, 23, 12, 21, 10, 24, 25, 11, 4, 16, 20, 7, 19]. The common theme in these works is to assume a certain *structure* on payoff functions. Assumptions of this type are natural in many applications, and often lead to efficient learning algorithms, e.g. see [18, 8] for a background.

In particular, the line of work [1, 17, 4, 20, 7, 19] considers the *Lipschitz MAB problem*, a broad and natural bandit setting in which the structure is induced by a metric on the set of arms.[1] In this setting an algorithm is given a metric space $(X, \mathcal{D})$, where $X$ is the set of arms, which represents the available *similarity information* (information on similarity between arms). Payoffs are *stochastic*: the payoff from choosing arm $x$ is an independent random sample with expectation $\mu(x)$. The metric space is related to payoffs via the following *Lipschitz condition*:[2]

$$|\mu(x) - \mu(y)| \leq \mathcal{D}(x, y) \quad \text{for all } x, y \in X. \tag{1}$$

Performance guarantees consider regret $R(t)$ as a function of time $t$, and focus on the asymptotical dependence of $R(\cdot)$ on a suitably defined "dimensionality" of the problem instance $(X, \mathcal{D}, \mu)$. Various upper and lower bounds of the form $R(t) = \tilde{\Theta}(t^\gamma)$, $\gamma < 1$ have been proved.

We relax an important assumption in Lipschitz MAB that the available similarity information provides numerical values in the sense of (1).[3] Specifically, following [21, 24, 25] we assume that an algorithm is (only) given a *taxonomy* on arms: a tree-based classification modeled by a rooted tree $\mathcal{T}$ whose leaf set is $X$. The idea is that any two arms in the same subtree are likely to have similar payoffs. Motivations include contextual advertising and web search with topical taxonomies, e.g. [25, 6, 29, 27], Monte-Carlo planning [21, 24], and Computer Go [13, 14].

We call the above formulation the *Taxonomy MAB problem*; a problem instance is a triple $(X, \mathcal{T}, \mu)$. Crucially, in Taxonomy MAB no ***numerical similarity information is explicitly revealed***. All prior algorithms for Lipschitz MAB (and in particular, all algorithms in [20, 7]) are parameterized by some numerical similarity information, and therefore do not directly apply to Taxonomy MAB.

One natural way to quantify the extent of similarity between arms in a given subtree is via the maximum difference in expected payoffs. Specifically, for each internal node $v$ we define the *width* of the corresponding subtree $\mathcal{T}(v)$ to be $\mathtt{W}(v) = \sup_{x,y \in X(v)} |\mu(x) - \mu(y)|$, where $X(v)$ is the set of leaves in $\mathcal{T}(v)$. Note that the subtree widths are non-increasing from root to leaves. A standard notion of distance induced by subtree widths, henceforth called *implicit distance*, is as follows: $\mathcal{D}_{\mathtt{imp}}(x, y)$ is the width of the least common ancestor of leaves $x, y$. It is immediate that this is indeed a metric, and moreover that it satisfies (1). In fact, $\mathcal{D}_{\mathtt{imp}}(x, y)$ is the smallest "width-based" distance that satisfies (1). If the widths are strictly decreasing, $\mathcal{T}$ can be reconstructed from $\mathcal{D}_{\mathtt{imp}}$.

Thus, an instance $(X, \mathcal{T}, \mu)$ of Taxonomy MAB naturally induces an instance $(X, \mathcal{D}_{\mathtt{imp}}, \mu)$ of Lipschitz MAB which (assuming the widths are strictly decreasing) encodes all relevant information. The crucial distinction is that in Taxonomy MAB the metric space $(X, \mathcal{D}_{\mathtt{imp}})$ is *implicit*: the subtree widths are not revealed to the algorithm. In particular, the algorithms in [20, 7] do not apply.

We view Lipschitz MAB as a performance benchmark for Taxonomy MAB. We are concerned with the following question: can an algorithm for Taxonomy MAB perform *as if* it was given the implicit metric space $(X, \mathcal{D}_{\mathtt{imp}})$? More formally, we ask whether it is possible to obtain guarantees for Taxonomy MAB that (almost) match the state-of-art for Lipschitz MAB.

We answer this question in the affirmative as long as the implicit metric space $(X, \mathcal{D}_{\mathtt{imp}})$ has a small doubling constant (see Section 2 for a milder condition). We provide an algorithm with guarantees that are almost identical to those for the *zooming algorithm* in [20].[4]

Our algorithm proceeds by estimating subtree widths of near-optimal subtrees. Thus, we encounter a two-pronged exploration-exploitation trade-off: samples from a given subtree reveal information not only about payoffs but also about the width, whereas in Lipschitz MAB we only need to worry about the payoffs. Dealing with this more complicated trade-off is the main new conceptual hurdle (which leads to some technical complications such as the proof of Lemma 4.4). These complications aside, our algorithm is similar to those in [17, 20] in that it maintains a partition of the space of arms into regions (in this case, subtrees) so that each region is treated as a "meta-arm", and this partition is adapted to the high-payoff regions.

## 1.1 Preliminaries

The *Taxonomy MAB problem* and the implicit metric space $(X, \mathcal{D}_{\texttt{imp}})$ are defined as in Section 1. We assume *stochastic payoffs* [2]: in each round $t$ the algorithm chooses a point $x = x_t \in X$ and observes an independent random sample from a payoff distribution $\mathcal{P}_{\texttt{payoff}}(x)$ with support $[0, 1]$ and expectation $\mu(x)$.[5] The *payoff function* $\mu : X \to [0, 1]$ is not revealed to an algorithm. The goal is to minimize *regret* with respect to the best expected arm:

$$R(T) \triangleq \mu^* T - \mathbb{E}\left[\sum_{t=1}^{T} \mu(x_t)\right] = \mathbb{E}\left[\sum_{t=1}^{T} \Delta(x_t)\right], \tag{2}$$

where $\mu^* \triangleq \sup_{x \in X} \mu(x)$ is the maximal expected payoff, and $\Delta(x) \triangleq \mu^* - \mu(x)$, is the "badness" of arm $x$. An arm $x \in X$ is called *optimal* if $\mu(x) = \mu^*$.

We will assume that the number of arms is finite (but possibly very large). Extension to infinitely many arms (which does not require new algorithmic ideas) is not included to simplify presentation. Also, we will assume a known time horizon (total number of rounds), denoted $T_{\text{hor}}$.

Our guarantees are in terms of the *zooming dimension* [20] of $(X, \mathcal{D}_{\texttt{imp}}, \mu)$, a concept that takes into account both the dimensionality of the metric space and the "goodness" of the payoff function. Below we specialize the definition from [20] to Taxonomy MAB.

**Definition 1.1** (zooming dimension)**.** For $X' \subset X$, define the *covering number* $N_\delta^{\text{cov}}(X')$ as the smallest number of subtrees of width at most $\delta$ that cover $X'$. Let $X_\delta \triangleq \{x \in X : 0 < \Delta(x) \le \delta\}$. The *zooming dimension* of a problem instance $\mathcal{I} = (X, \mathcal{T}, \mu)$, with multiplier $c$, is

$$\texttt{ZoomDim}(\mathcal{I}, c) \triangleq \inf\{d \ge 0 : N_{\delta/8}^{\text{cov}}(X_\delta) \le c\,\delta^{-d} \quad \forall \delta > 0\}. \tag{3}$$

In other words, we consider a covering property $N_{\delta/8}^{\text{cov}}(X_\delta) \le c\,\delta^{-d}$, and define the zooming dimension as the smallest $d$ such that this covering property holds for all $\delta > 0$. The zooming dimension essentially coincides with the covering dimension of $(X, \mathcal{D})$ [6] for the worst-case payoff function $\mu$, but can be (much) smaller when $\mu$ is "benign". In particular, zooming dimension would "ignore" a subtree with high covering dimension but significantly sub-optimal payoffs.

The *doubling constant* $c_{\text{DBL}}$ of a metric space is the smallest $k$ such that any ball can be covered by $k$ balls of half the radius. (In our case, any subtree can be covered by $k$ subtrees of half the width.) Doubling constant has been a standard notion in theoretical computer science literature since [15]; since then, it was used to characterize tractable problem instances for a variety of problems. It is known that $c_{\text{DBL}} = O(2^d)$ for any bounded subset $S \subset \mathbb{R}^{d'}$ of linear dimension $d$, under any metric $\ell_p, p \ge 1$. Moreover, $c_{\text{DBL}} \ge c\,2^d$ if $d$ is the covering dimension with multiplier $c$.

## 2 Statement of results

We will prove that our algorithm (`TaxonomyZoom`) satisfies the following regret bound:

For each instance $\mathcal{I}$ of Taxonomy MAB, each $c > 0$ and each $T \le T_{\text{hor}}$,

$$R(T) \le O(c\,K_{\mathcal{I}} \log T_{\text{hor}})^{1/(2+d)} \times T^{1-1/(2+d)}, \quad d = \texttt{ZoomDim}(\mathcal{I}, c). \tag{4}$$

We will bound the factor $K_{\mathcal{I}}$ below. For $K_{\mathcal{I}} = 1$ this is the guarantee for the zooming algorithm in [20] for the corresponding instance $(X, \mathcal{D}_{\texttt{imp}}, \mu)$ of Lipschitz MAB. Note that the definition of zooming dimension allows a trade-off between $c$ and $d$, and we obtain the optimal trade-off since (4) holds for all values of $c$ at once. Following the prior work on Lipschitz MAB, we identify the exponent in (4) as the crucial parameter, as long as the multiplier $c$ is sufficiently small.[7]

Our first (and crude) bound for $K_{\mathcal{I}}$ is in terms of the doubling constant of $(X, \mathcal{D}_{\texttt{imp}})$.

**Theorem 2.1** (Crude bound)**.** *Given an upper bound $c'_{\text{DBL}}$ on the doubling constant of $(X, \mathcal{D}_{\texttt{imp}})$,* `TaxonomyZoom` *achieves* (4) *with $K_{\mathcal{I}} = f(c'_{\text{DBL}}) \log |X|$, where $f(n) = n\,2^n$.*

Our main result (which implies Theorem 2.1) uses a more efficient bound for $K_{\mathcal{I}}$.

Recall that in Taxonomy MAB subtree widths are not revealed, and the algorithm has to use sampling to estimate them. Informally, the taxonomy is useful for our purposes if and only if subtree widths can be efficiently estimated using random sampling. We quantify this as a parameter called `quality`, and bound $K_{\mathcal{I}}$ in terms of this parameter.

We use simple random sampling: start at a tree node $v$ and choose a branch uniformly at random at each junction. Let $\mathcal{P}(u|v)$ be the probability that node $u$ is reached starting from $v$. The probabilities $\mathcal{P}(\cdot|v)$ induce a distribution on $X(v)$, the leaf set of subtree $\mathcal{T}(v)$. A sample from this distribution is called a *random sample* from $\mathcal{T}(v)$, with expected payoff $\mu(v) \triangleq \sum_{x \in X(v)} \mu(x)\, \mathcal{P}(x|v)$.

**Definition 2.2.** The `quality` of the taxonomy for a given problem instance is the largest number $q \in (0,1)$ with the following property: for each subtree $\mathcal{T}(v)$ containing an optimal arm there exist tree nodes $u, u' \in \mathcal{T}(v)$ such that $\mathcal{P}(u|v)$ and $\mathcal{P}(u'|v)$ are at least $q$ and

$$|\mu(u) - \mu(u')| \geq \tfrac{1}{2}\, \mathtt{W}(v). \tag{5}$$

One could use the pair $u, u'$ in Definition 2.2 to obtain reliable estimates for $\mathtt{W}(v)$. The definition focuses on the difficulty of obtaining such pair via random sampling from $\mathcal{T}(v)$. The definition is flexible: it allows $u$ and $u'$ to be at different depth (which is useful if node degrees are large and non-uniform) and the widths of other internal nodes in $\mathcal{T}(v)$ cannot adversely impact `quality`. The constant $\tfrac{1}{2}$ in (5) is arbitrary; we fix it for convenience.

For a particularly simple example, consider a binary taxonomy such that for each subtree $\mathcal{T}(v)$ containing an optimal arm there exist grandchildren $u, u'$ of $v$ that satisfy (5). For instance, such $u, u'$ exist if the width of each grandchild of $v$ is at most $\tfrac{1}{4}\, \mathtt{W}(v)$. Then `quality` $\geq \tfrac{1}{4}$.

**Theorem 2.3** (Main result). *Assume an lower bound $q \leq$ `quality`$(\mathcal{I})$ is known. Then* `TaxonomyZoom` *achieves* (4) *with $K_{\mathcal{I}} = \frac{\mathtt{deg}}{q} \log |X|$, where* `deg` *is the degree of the taxonomy.*

Theorem 2.1 follows because, letting $c_{\mathrm{DBL}}$ be the doubling constant of $(X, \mathcal{D}_{\mathtt{imp}})$, all node degrees are at most $c_{\mathrm{DBL}}$ and moreover `quality` $\geq 2^{-c_{\mathrm{DBL}}}$ (we omit the proof from this version).

**Discussion.** The guarantee in Theorem 2.3 is instance-dependent: it depends on `deg`/`quality` and `ZoomDim`, and is meaningful only if these quantities are small compared to the number of arms (informally, we will call such problem instances "benign"). Also, the algorithm needs to know a non-trivial lower bound on `quality`; very conservative estimates would not suffice. However, underestimating `quality` (and likewise overestimating $T_{\mathrm{hor}}$) is relatively inexpensive as long as the "influence" of these parameters on regret is eventually dominated by the $T^{1-1/(2+d)}$ term.

For benign problem instances, the benefit of using the taxonomy is the vastly improved dependence on the number of arms. Without a taxonomy or any other structure, regret of any algorithm for stochastic MAB scales *linearly* in the number of (near-optimal) arms, for a sufficiently large $t$. Specifically, let $N_\delta$ be the number of arms $x$ such that $\frac{\delta}{2} < \Delta(x) \leq \delta$. Then the worst-case regret (over all problem instances) cannot be better than $R(t) = \min(\delta t, \Omega(\frac{1}{\delta}\, N_\delta))$. [8]

An alternative approach to MAB problems on trees (without knowing the "widths") are the "tree bandit algorithms" explored in [21, 24]. Here for each tree node $v$ there is a separate, independent copy of UCB1 [2] or a UCB1-style index algorithm (call it $\mathcal{A}_v$), so that the "arms" for $\mathcal{A}_v$ correspond to children $u$ of $v$, and selecting a child $u$ in a given round corresponds to playing $\mathcal{A}_u$ in this round. [21, 24] report successful empirical performance of such algorithms on some examples. However, regret bounds for these algorithms do not scale as well with the number of arms: even if the tree widths are given, then letting $\Delta_{\min} \triangleq \min_{x \in X : \Delta(x) > 0} \Delta(x)$, the regret bound is proportional to $|X_\delta|/\Delta_{\min}$ (where $X_\delta$ is as in Definition 1.1), whereas the regret bound in Theorem 2.3 is (essentially) in terms of the covering numbers $N^{\mathrm{cov}}_{\delta/8}(X_\delta)$.

# 3   Main algorithm

Our algorithm $\mathtt{TaxonomyZoom}(T_{\mathrm{hor}}, q)$ is parameterized by the time horizon $T_{\mathrm{hor}}$ and the *quality parameter* $q \leq \mathtt{quality}$. In each round the algorithm *selects* one of the tree nodes, say $v$, and plays a randomly sampled arm $x$ from $\mathcal{T}(v)$. We say that a subtree $\mathcal{T}(u)$ is *hit* in this round if $u \in \mathcal{T}(v)$ and $x \in \mathcal{T}(u)$. For each tree node $v$ and time $t$, let $n_t(v)$ be the number of times the subtree $\mathcal{T}(v)$ has been hit by the algorithm before time $t$, and let $\mu_t(v)$ be the corresponding average reward. Note that $E[\mu_t(v) \,|\, n_t(v) > 0] = \mu(v)$. Define the *confidence radius* of $v$ at time $t$ as

$$\mathtt{rad}_t(v) \triangleq \sqrt{8 \, \log(T_{\mathrm{hor}} |X|) \,/\, (2 + n_t(v))}. \tag{6}$$

The meaning of the confidence radius is that $|\mu_t(v) - \mu(v)| \leq \mathtt{rad}_t(v)$ with high probability.

For each tree node $v$ and time $t$, let us define the *index* of $v$ at time $t$ as

$$I_t(v) \triangleq \mu_t(v) + (1 + 2 \, k_{\mathcal{A}}) \, \mathtt{rad}_t(v), \quad \text{where} \quad k_{\mathcal{A}} \triangleq 4\sqrt{2/q}. \tag{7}$$

Here we posit $\mu_t(v) = 0$ if $n_t(v) = 0$. Let us define the width estimate[9]

$$\mathtt{W}_t(v) \triangleq \max(0, \, U_t(v) - L_t(v)), \text{ where } \begin{cases} U_t(v) \triangleq \max_{u \in \mathcal{T}(v), \, s \leq t} \; \mu_s(u) - \mathtt{rad}_s(u), \\ L_t(v) \triangleq \min_{u \in \mathcal{T}(v), \, s \leq t} \; \mu_s(u) + \mathtt{rad}_s(u). \end{cases} \tag{8}$$

Here $U_t(v)$ is the best available lower confidence bound on $\max_{x \in X(v)} \mu(x)$, and $L_t(v)$ is the best available upper confidence bound on $\min_{x \in X(v)} \mu(x)$. If both bounds hold then $\mathtt{W}_t(v) \leq \mathtt{W}(v)$.

Throughout the phase, some tree nodes are designated *active*. We maintain the following invariant:

$$\mathtt{W}_t(v) < k_{\mathcal{A}} \, \mathtt{rad}_t(v) \; \text{ for each active internal node } v. \tag{9}$$

$\mathtt{TaxonomyZoom}(T_{\mathrm{hor}}, q)$ operates as follows. Initially the only active tree node is the root. In each round, the algorithm performs the following three steps:

(S1)  While Invariant (9) is violated by some $v$, de-activate $v$ and activate all its children.
(S2)  Select an active tree node $v$ with the maximal index (7), breaking ties arbitrarily.
(S3)  Play a randomly sampled arm from $\mathcal{T}(v)$.

Note that each arm is activated and deactivated at most once.

**Implementation details.**   If an explicit representation of the taxonomy can be stored in memory, then the following simple implementation is possible. For each tree node $v$, we store several statistics: $n_t$, $\mu_t$, $U_t$ and $L_t$. Further, we maintain a linked list of active nodes, sorted by the index. Suppose in a given round $t$, a subtree $v$ is chosen, and an arm $x$ is played. We update the statistics by going up the $x \to v$ path in the tree (note that only the statistics on this path need to be updated). This update can be done in time $O(\mathtt{depth}(x))$. Then one can check whether Invariant (9) holds for a given node in time $O(1)$. So step (S1) of the algorithm can be implemented in time $O(1 + N)$, where $N$ is the number of nodes activated during this step. Finally, the linked list of active nodes can be updated in time $O(1 + N)$. Then the selections in steps (S2) and (S3) are done in time $O(1)$.

**Lemma 3.1.** $\mathtt{TaxonomyZoom}$ *can be implemented with $O(1)$ storage per each tree node, so that in each round the time complexity is $O(N + \mathtt{depth}(x))$, where $N$ is the number of arms activated in step (S1), and $x$ is the arm chosen in step (S3).*

Sometimes it may be feasible (and more space-efficient) to represent the taxonomy implicitly, so that a tree node is expanded only if needed. Specifically, suppose the following interface is provided: given a tree node $v$, return all its children and an arbitrary arm $x \in \mathcal{T}(v)$. Then $\mathtt{TaxonomyZoom}$ can be implemented so that it only stores the statistics for each node $u$ such that $\mathcal{P}(u|v) \geq q$ for some active node $v$ (rather than for all tree nodes).[10] The running times are as in Lemma 3.1.

# 4  Analysis: proof of Theorem 2.3

First, let us fix some notation. We will focus on regret up to a fixed time $T \leq T_{\text{hor}}$. In what follows, let $d = \texttt{ZoomDim}(\mathcal{I}, c)$ for some fixed $c > 0$. Recall the notation $X_\delta \triangleq \{x \in X : \Delta(x) \leq \delta\}$ from Definition 1.1. Here $\delta$ is the " distance scale"; we will be interested in $\delta \geq \delta_0$, for

$$\delta_0 \triangleq (\tfrac{K}{T})^{1/(d+2)}, \text{ where } K \triangleq O(c \deg k_{\mathcal{A}}^2 \log T_{\text{hor}}). \tag{10}$$

We identify a certain high-probability behavior of the algorithm, and argue deterministically conditional on the event that this behavior actually holds.

**Definition 4.1.** An execution of TaxonomyZoom is called *clean* if for each time $t \leq T$ and all tree nodes $v$ the following two properties hold:

(P1)  $|\mu_t(v) - \mu(v)| \leq \texttt{rad}_t(v)$ as long as $n_t(v) > 0$.
(P2)  If $u \in \mathcal{T}(v)$ then

$$n_t(v) \, \mathcal{P}(u|v) \geq 8 \log T \quad \Rightarrow \quad n_t(u) \geq \tfrac{1}{2} \, n_t(v) \, \mathcal{P}(u|v).$$

Note that in a clean execution the quantities in (8) satisfy the desired high-confidence bounds: $U_t(v) \leq \max_{x \in X(v)} \mu(x)$ and $L_t(v) \geq \min_{x \in X(v)} \mu(x)$, which implies $\mathtt{W}(v) \geq \mathtt{W}_t(v)$.

**Lemma 4.2.** *An execution of* TaxonomyZoom *is clean with probability at least* $1 - 2\,T_{\text{hor}}^{-2}$.

*Proof.* For part (P1), fix a tree node $v$ and let $\zeta_j$ to be the payoff in the $j$-th round that $v$ has been hit. Then $\{\sum_{j=1}^n (\zeta_j - \mu(v))\}_{n=1..T}$ is a martingale.[11] Let $\bar{\zeta}_n \triangleq \frac{1}{n} \sum_{j=1}^n \zeta_j$ be the $n$-th average. Then by the Azuma-Hoeffding inequality, for any $n \leq T_{\text{hor}}$ we have:

$$\Pr[\,|\bar{\zeta}_n - \mu(v)| > r(n)] \leq (T_{\text{hor}} \, |X|)^{-2}, \text{ where } r(n) \triangleq \sqrt{8 \, \log(T_{\text{hor}}|X|) \, / \, (2 + n)}. \tag{11}$$

Note that $\texttt{rad}_t(v) = r(n_t(v))$. We obtain (P1) by taking the Union Bound for (11) over all nodes $v$ and all $n \leq T$. (This is the only place where we use the $\log |X|$ term in (6).)

Part (P2) is proved via a similar application of martingales and Azuma-Hoeffding inequality. $\quad\square$

From now on we will argue about a clean execution. Recall that by definition of $\mathtt{W}(\cdot)$,

$$\mu(v) \leq \mu(u) + \mathtt{W}(v) \quad \text{for any tree node } u \in \mathcal{T}(v). \tag{12}$$

The crux of the proof of Theorem 2.3 is that at all times the maximal index is at least $\mu^*$.

**Lemma 4.3.** *Consider a clean execution of* TaxonomyZoom$(T_{\text{hor}}, q)$. *Then the following holds: in any round* $t \leq T_{\text{hor}}$, *at any point in the execution such that the invariant* (9) *holds, there exists an active tree node* $v^*$ *such that* $I_t(v^*) \geq \mu^*$.

*Proof.* Fix an optimal arm $x^* \in X$. Note that in each round $t$, there exist an active tree node $v_t^*$ such that $x^* \in \mathcal{T}(v^*)$. (One can prove it by induction on $t$, using the (de)activation rule (S1) in TaxonomyZoom.) Fix round $t$ and the corresponding tree node $v^* = v_t^*$.

By Definition 2.2, there exist $v_0, v_1 \in \mathcal{T}_q(v^*)$ such that $|\mu(v_1) - \mu(v_0)| \geq \mathtt{W}(v^*)/2$.

Assume that $\Delta \triangleq \mathtt{W}(v^*) > 0$, and define $f(\Delta) = 8^3 \log(T_{\text{hor}}) \Delta^{-2}$. Then for each tree node $v$

$$\texttt{rad}_t(v) \leq \Delta/8 \iff n_t(v) \geq f(\Delta). \tag{13}$$

Now, for the sake of contradiction let us suppose that $n_t(v^*) \geq (\tfrac{1}{4} k_{\mathcal{A}})^2 f(\Delta)$. By (13), this is equivalent to $\Delta \geq 2 k_{\mathcal{A}} \texttt{rad}_t(v^*)$. Note that $n_t(v^*) \geq (2/q) f(\Delta)$ by our assumption on $k_{\mathcal{A}}$, so by property (P2) in the definition of the clean execution, for each node $v_j$, $j \in \{0, 1\}$ we have $n_t(v_j) \geq f(\Delta)$, which implies $\texttt{rad}_t(v_j) \leq \Delta/8$. Therefore (8) gives a good estimate of $\mathtt{W}(v^*)$, namely $\mathtt{W}_t(v^*) \geq \Delta/4$. It follows that $\mathtt{W}_t(v^*) \geq k_{\mathcal{A}} \texttt{rad}_t(v^*)$, which violates Invariant (9).

We proved that $\mathtt{W}(v^*) \leq 2 k_{\mathcal{A}} \texttt{rad}_t(v^*)$. Using (12), we have $\Delta(v^*) \leq \mathtt{W}(v^*) < 2 k_{\mathcal{A}} \texttt{rad}_t(v^*)$ and

$$I_t(v^*) \geq \mu(v^*) + 2 k_{\mathcal{A}} \texttt{rad}_t(v^*) \geq \mu^*, \tag{14}$$

where the first inequality in (14) holds by definition (7) and property (P1) of a clean execution. $\quad\square$

We use Lemma 4.3 to show that the algorithm does not activate too many tree nodes with large badness $\Delta(\cdot)$, and each such node is not played too often. For each tree node $v$, let $N(v)$ be the number of times node $v$ was selected in step (S2) of the algorithm. Call $v$ *positive* if $N(v) > 0$. We partition all positive tree nodes and all deactivated tree nodes into sets

$$S_i = \{\text{positive tree nodes } v : 2^{-i} < \Delta(v) \leq 2^{-i+1}\},$$
$$S_i^* = \{\text{deactivated tree nodes } v : 2^{-i} < 4\,\mathtt{W}(v) \leq 2^{-i+1}\}.$$

**Lemma 4.4.** *Consider a clean execution of algorithm* $\mathtt{TaxonomyZoom}(T_{\mathrm{hor}}, q)$.

- (a) *for each tree node $v$ we have $N(v) \leq O(k_{\mathcal{A}}^2 \log T_{\mathrm{hor}})\,\Delta^{-2}(v)$.*
- (b) *if node $v$ is de-activated at some point in the execution, then $\Delta(v) \leq 4\,\mathtt{W}(v)$.*
- (c) *For each $i$, $|S_i^*| \leq 2K_i$, where $K_i \triangleq c\, 2^{(i+1)\,d}$.*
- (d) *For each $i$, $|S_i| \leq O(\mathtt{deg}\, K_{i+1})$.*

*Proof.* For part (a), fix an arbitrary tree node $v$ and let $t$ be the last time $v$ was selected in step (S2) of the algorithm. By Lemma 4.3, at that point in the execution there was a tree node $v^*$ such that $I_t(v^*) \geq \mu^*$. Then using the selection rule (step (S2)) and the definition of index (7), we have

$$\mu^* \leq I_t(v^*) \leq I_t(v) \leq \mu(v) + (2 + 2\,k_{\mathcal{A}})\,\mathtt{rad}_t(v),$$
$$\Delta(v) \leq (2 + 2\,k_{\mathcal{A}})\,\mathtt{rad}_t(v). \tag{15}$$
$$N(v) \leq n_t(v) \leq O(k_{\mathcal{A}}^2 \log T_{\mathrm{hor}})\,\Delta^{-2}(v).$$

**For part (b),** suppose tree node $v$ was de-activated at time $s$. Let $t$ be the last round in which $v$ was selected. Then

$$\mathtt{W}(v) \geq \mathtt{W}_s(v) \geq k_{\mathcal{A}}\, r_s(v) \geq \tfrac{1}{3}\,(2 + 2\,k_{\mathcal{A}})\,\mathtt{rad}_t(v) \geq \tfrac{1}{3}\,\Delta(v). \tag{16}$$

Indeed, the first inequality in (16) holds since we are in a clean execution, the second inequality in (16) holds because $v$ was de-activated, the third inequality holds because $n_s(v) = n_t(v) + 1$, and the last inequality in (16) holds by (15).

**For part (c),** let us fix $i$ and define $Y_i = \{x \in X : \Delta(x) \leq 2^{-i+1}\}$. By Definition 1.1, this set can be covered by $K_i$ subtrees $\mathcal{T}(v_1), \ldots, \mathcal{T}(v_{K_i})$, each of width $< 2^{-i}/4$. Fix a deactivated tree node $v \in S_i^*$. For each arm $x \in X$ in subtree $\mathcal{T}(v)$ we have, by part (b),

$$\Delta(x) \leq \Delta(v) + \mathtt{W}(v) \leq 4\,\mathtt{W}(v) \leq 2^{-i+1},$$

so $x \in Y_i$ and therefore is contained in some $\mathcal{T}(v_j)$. Note that $v_j \in \mathcal{T}(v)$ since $\mathtt{W}(v) > \mathtt{W}(v_j)$. It follows that the subtrees $\mathcal{T}(v_1), \ldots, \mathcal{T}(v_K)$ cover the leaf set of $\mathcal{T}(v)$.

Consider the graph $G$ on the node set $S_i^* \cup \{v_1, \ldots, v_K\}$, where two nodes $u, v$ are connected by a directed edge $(u, v)$ if there is a path from $u$ to $v$ in the tree $\mathcal{T}$. This is a directed forest of out-degree at least 2, whose leaf set is a subset of $\{v_1, \ldots, v_{K_i}\}$. Since in any directed tree of out-degree $\geq 2$ the number of nodes is at most twice the number of leaves, $G$ contains at most $K_i$ internal nodes. Thus $|S_i^*| \leq 2K_i$, proving part (c).

**For part (d),** let us fix $i$ and consider a positive tree node $u \in S_i$. Since $N(u) > 0$, either $u$ is active at time $T_{\mathrm{hor}}$, or it was deactivated in some round before $T_{\mathrm{hor}}$. In the former case, let $v$ be the parent of $u$. In the latter case, let $v = u$. Then by part (b) we have $2^{-i} \leq \Delta(u) \leq \Delta(v) + \mathtt{W}(v) \leq 4\,\mathtt{W}(v)$, so $v \in S_j^*$ for some $j \leq i + 1$.

For each tree node $v$, define its *family* as the set which consists of $u$ itself and all its children. We have proved that each positive node $u \in S_i$ belongs to the family of some deactivated node $v \in \cup_{j=1}^{i+1} S_j^*$. Since each family consists of at most $1 + \mathtt{deg}$ nodes, it follows that

$$|S_i| \leq (1 + \mathtt{deg})\,(\textstyle\sum_{j=1}^{i+1} K_j) \leq O(\mathtt{deg}\, K_{i+1}). \quad \square$$

**Proof of Theorem 2.3:** The theorem follows Lemma 4.4(ad). Let us assume a clean execution. (Recall that by Lemma 4.2 the failure probability is sufficiently small to be neglected.) Then:

$$\textstyle\sum_{v \in S_i} N(v)\,\Delta(v) \leq O(k_{\mathcal{A}}^2 \log T_{\mathrm{hor}}) \sum_{v \in S_i} \frac{1}{\Delta(v)} \leq O(k_{\mathcal{A}}^2 \log T_{\mathrm{hor}})\,|S_i|\,2^i \leq K\,2^{(i+2)(1+d)},$$

where $K$ is defined in (10). For any $\delta_0 = 2^{-i_0}$ we have

$$R(T) \leq \sum\nolimits_{\text{tree nodes } v} N(v)\,\Delta(v)$$

$$= \left(\sum\nolimits_{v:\,\Delta(v)<\delta_0} N(v)\,\Delta(v)\right) + \left(\sum\nolimits_{v:\,\Delta(v)\geq\delta_0} N(v)\,\Delta(v)\right)$$

$$\leq \delta_0 T + \left(\sum\nolimits_{i\leq i_0}\sum\nolimits_{v\in S_i} N(v)\,\Delta(v)\right) \leq \delta_0 T + \sum\nolimits_{i\leq i_0} K\,2^{(i+2)(1+d)}$$

$$\leq \delta_0 T + O(K)\,(\tfrac{8}{\delta_0})^{(1+d)}.$$

We obtain the desired regret bound (4) by setting $\delta_0$ as in (10). $\qquad\square$

## 5    (De)parameterizing the algorithm

Recall that `TaxonomyZoom` needs to be parameterized by $T_{\text{hor}}$ and $q$. dependence on the parameters can be removed using a suitable version of the standard *doubling trick*: consider a "meta-algorithm" that proceeds in phases so that in each phase $i = 1, 2, 3, \ldots$ a fresh instance of `TaxonomyZoom`$(2^i, q_i)$ is run for $2^i$ rounds, where $q_i$ slowly decreases with $i$. For instance, if we take $q_i = 2^{-\alpha i}$ for some $\alpha \in (0, 1)$ then this meta-algorithm has regret

$$R(T) \leq O(c\,\texttt{deg}\,\log T)^{1/(2+d)} \times T^{1-(1-\alpha)/(2+d)} \qquad \forall T \geq \texttt{quality}^{-1/\alpha} \tag{17}$$

where $d = \texttt{ZoomDim}(\mathcal{I}, c)$, for any given $c > 0$.

While the doubling trick is very useful in theory of online decision problems, its practical importance is questionable, as running a fresh algorithm instance in each phase seems unnecessarily wasteful. We conjecture that in practice one could run a single instance of the algorithm while gradually increasing $T_{\text{hor}}$ and decreasing $q$. However, providing provable guarantees for this modified algorithm seems beyond the current techniques. In particular, extending a much simpler analysis of the zooming algorithm [20] to arbitrary time horizon remains a challenge.[12]

Further, we conjecture that `TaxonomyZoom` will typically work in practice even if the parameters are misspecified, i.e. even if $T_{\text{hor}}$ is too low and $q$ is too optimistic. Indeed, recall that our algorithm is *index-based*, in the style of UCB1 [2]. The only place where the parameters are invoked is in the definition of the index (7), namely in the constant in front of the exploration term. It has been observed in [28, 29] that in a related MAB setting, reducing this constant to 1 from the theoretically mandated $\Theta(\log T)$-type term actually improves algorithms' performance in simulations.

## 6    Conclusions

In this paper, we have extended previous multi-armed bandit learning algorithms with large numbers of available strategies. Whereas the most effective previous approaches rely on explicitly knowing the distance between available strategies, we consider the case where the distances are implicit in a hierarchy of available strategies. We have provided a learning algorithm for this setting, and show that its performance almost matches the best known guarantees for the Lipschitz MAB problem. Further, we have shown how our approach results in stronger provable guarantees than alternative algorithms such as tree bandit algorithms [21, 24].

We conjecture that the dependence on `quality` (or some version thereof) is necessary for the worst-case regret bounds, even if `ZoomDim` is low. It is an open question whether there are non-trivial families of problem instances with low `quality` for which one could achieve low regret.

Our results suggest some natural extensions. Most interestingly, a number of applications recently posed as MAB problems over large sets of arms – including learning to rank online advertisements or web documents (e.g. [26, 29]) – naturally involve choosing among arms (e.g. ads) that can be classified according to any of a number of hierarchies (e.g. by class of product sold, geographic location, etc). In particular, such different hierarchies may be of different usefulness. Selecting among, or combining from, a set of available hierarchical representations of arms poses interesting challenges. More generally, we would like to generalize Theorem 2.3 to other structures that implicitly define a metric space on arms (in the sense of (1)). One specific target would be directed acyclic graphs. While our algorithm is well-defined for this setting, the theoretical analysis does not apply.

## Footnotes

[1]This problem has been explicitly defined in [20]. Preceding work [1, 17, 9, 4] considered a few special cases such as a one-dimensional real interval with a metric defined by $\mathcal{D}(x, y) = |x - y|^\alpha$, $\alpha \in (0, 1]$.

[2]Lipschitz constant is $c_{\mathtt{lip}} = 1$ without loss of generality: else, one could take a metric $c_{\mathtt{lip}} \times \mathcal{D}$.

[3]In the full version of [20] the setting is relaxed so that (1) needs to hold only if $x$ is optimal, and the distances between non-optimal points do not need to be explicitly known; [7] provides a similar result.

[4]The guarantees in [7] are similar but slightly different technically.

[5]Other than support and expectation, the "shape" of $\mathcal{P}_{\texttt{payoff}}(x)$ is not essential for this paper.

[6]Covering dimension is defined as in (3), replacing $N_{\delta/8}^{\text{cov}}(X_\delta)$ with $N_\delta^{\text{cov}}(X)$..

[7]One can reduce `ZoomDim` by making $c$ huge, e.g. `ZoomDim` $= 0$ for $c = |X|$. However, this is not likely to lead to useful regret bounds. Similar trade-off (dimension vs multiplier) is implicit in [7].

[8]This is implicit from the lower-bounding analysis in [22] and [3].

[9] Defining $U_t$, $L_t$ in (8) via $s \leq t$ (rather than $s = t$) improves performance, but is not essential for analysis.

[10] The algorithm needs to be modified slightly; we leave the details to the full version.

[11]To make $\zeta_n$ well-defined for any $n \leq T_{\text{hor}}$, consider a hypothetical algorithm which coincides with TaxonomyZoom for the first $T_{\text{hor}}$ rounds and then proceeds so that each tree node is selected $T_{\text{hor}}$ times.

[12] However, [7] obtains similar guarantees for arbitrary time horizon, with a different algorithm.

# References

[1] Rajeev Agrawal. The continuum-armed bandit problem. *SIAM J. Control and Optimization*, 33(6):1926–1951, 1995.

[2] Peter Auer, Nicolò Cesa-Bianchi, and Paul Fischer. Finite-time analysis of the multiarmed bandit problem. *Machine Learning*, 47(2-3):235–256, 2002. Preliminary version in *15th ICML*, 1998.

[3] Peter Auer, Nicolò Cesa-Bianchi, Yoav Freund, and Robert E. Schapire. The nonstochastic multiarmed bandit problem. *SIAM J. Comput.*, 32(1):48–77, 2002. Preliminary version in *36th IEEE FOCS*, 1995.

[4] Peter Auer, Ronald Ortner, and Csaba Szepesvári. Improved Rates for the Stochastic Continuum-Armed Bandit Problem. In *20th COLT*, pages 454–468, 2007.

[5] Baruch Awerbuch and Robert Kleinberg. Online linear optimization and adaptive routing. *J. of Computer and System Sciences*, 74(1):97–114, February 2008. Preliminary version in *36th ACM STOC*, 2004.

[6] Andrei Broder, Marcus Fontoura, Vanja Josifovski, and Lance Riedel. A semantic approach to contextual advertising. In *30th SIGIR*, pages 559–566, 2007.

[7] Sébastien Bubeck, Rémi Munos, Gilles Stoltz, and Csaba Szepesvari. Online Optimization in X-Armed Bandits. *J. of Machine Learning Research (JMLR)*, 12:1587–1627, 2011. Preliminary version in *NIPS 2008*.

[8] Nicolò Cesa-Bianchi and Gábor Lugosi. *Prediction, learning, and games*. Cambridge Univ. Press, 2006.

[9] Eric Cope. Regret and convergence bounds for immediate-reward reinforcement learning with continuous action spaces. *IEEE Trans. on Automatic Control*, 54(6):1243–1253, 2009. A manuscript from 2004.

[10] Varsha Dani and Thomas P. Hayes. Robbing the bandit: less regret in online geometric optimization against an adaptive adversary. In *17th ACM-SIAM SODA*, pages 937–943, 2006.

[11] Varsha Dani, Thomas P. Hayes, and Sham Kakade. The Price of Bandit Information for Online Optimization. In *20th NIPS*, 2007.

[12] Abraham Flaxman, Adam Kalai, and H. Brendan McMahan. Online Convex Optimization in the Bandit Setting: Gradient Descent without a Gradient. In *16th ACM-SIAM SODA*, pages 385–394, 2005.

[13] Sylvain Gelly and David Silver. Combining online and offline knowledge in UCT. In *24th ICML*, 2007.

[14] Sylvain Gelly and David Silver. Achieving master level play in 9x9 computer go. In *23rd AAAI*, 2008.

[15] Anupam Gupta, Robert Krauthgamer, and James R. Lee. Bounded geometries, fractals, and low–distortion embeddings. In *44th IEEE FOCS*, pages 534–543, 2003.

[16] Sham M. Kakade, Adam T. Kalai, and Katrina Ligett. Playing Games with Approximation Algorithms. In *39th ACM STOC*, 2007.

[17] Robert Kleinberg. Nearly tight bounds for the continuum-armed bandit problem. In *18th NIPS*, 2004.

[18] Robert Kleinberg. *Online Decision Problems with Large Strategy Sets*. PhD thesis, MIT, 2005.

[19] Robert Kleinberg and Aleksandrs Slivkins. Sharp Dichotomies for Regret Minimization in Metric Spaces. In *21st ACM-SIAM SODA*, 2010.

[20] Robert Kleinberg, Aleksandrs Slivkins, and Eli Upfal. Multi-Armed Bandits in Metric Spaces. In *40th ACM STOC*, pages 681–690, 2008.

[21] Levente Kocsis and Csaba Szepesvari. Bandit Based Monte-Carlo Planning. In *17th ECML*, pages 282–293, 2006.

[22] T.L. Lai and Herbert Robbins. Asymptotically efficient Adaptive Allocation Rules. *Advances in Applied Mathematics*, 6:4–22, 1985.

[23] H. Brendan McMahan and Avrim Blum. Online Geometric Optimization in the Bandit Setting Against an Adaptive Adversary. In *17th COLT*, pages 109–123, 2004.

[24] Rémi Munos and Pierre-Arnaud Coquelin. Bandit algorithms for tree search. In *23rd UAI*, 2007.

[25] Sandeep Pandey, Deepak Agarwal, Deepayan Chakrabarti, and Vanja Josifovski. Bandits for Taxonomies: A Model-based Approach. In *SDM*, 2007.

[26] Sandeep Pandey, Deepayan Chakrabarti, and Deepak Agarwal. Multi-armed Bandit Problems with Dependent Arms. In *24th ICML*, 2007.

[27] Susan T. Dumais Paul N. Bennett, Krysta Marie Svore. Classification-enhanced ranking. In *19th WWW*, pages 111–120, 2010.

[28] Filip Radlinski, Robert Kleinberg, and Thorsten Joachims. Learning diverse rankings with multi-armed bandits. In *25th ICML*, pages 784–791, 2008.

[29] Aleksandrs Slivkins, Filip Radlinski, and Sreenivas Gollapudi. Learning optimally diverse rankings over large document collections. In *27th ICML*, pages 983–990, 2010.

